# A Neural Network for Motion Detection of Drift-Balanced Stimuli

**Hilary Tunley\***
School of Cognitive and Computer Sciences
Sussex University
Brighton, England.

## Abstract

This paper briefly describes an artificial neural network for preattentive visual processing. The network is capable of determining image motion in a type of stimulus which defeats most popular methods of motion detection – a subset of second-order visual motion stimuli known as drift-balanced stimuli(DBS). The processing stages of the network described in this paper are integratable into a model capable of simultaneous motion extraction. edge detection, and the determination of occlusion.

## 1 INTRODUCTION

Previous methods of motion detection have generally been based on one of two underlying approaches: correlation; and gradient-filter. Probably the best known example of the correlation approach is the Reichardt movement detector [Reichardt 1961]. The gradient-filter (GF) approach underlies the work of Adelson and Bergen [Adelson 1985], and Heeger [Heeger 1988], amongst others.

These motion-detecting methods cannot track DBS, because DBS lack essential components of information needed by such methods. Both the correlation and GF approaches impose constraints on the input stimuli. Throughout the image sequence, correlation methods require information that is spatiotemporally correlatable; and GF motion detectors assume temporally constant spatial gradients.

The network discussed here does not impose such constraints. Instead, it extracts motion *energy* and exploits the spatial coherence of movement (defined more formally in the Gestalt theory of *common fate* [Koffka 1935]) to achieve tracking.

The remainder of this paper discusses DBS image sequences, then correlation methods, then GF methods in more detail, followed by a qualitative description of this network which *can* process DBS.

## 2   SECOND-ORDER AND DRIFT-BALANCED STIMULI

There has been a lot of recent interest in second-order visual stimuli, and DBS in particular ([Chubb 1989, Landy 1991]). DBS are stimuli which give a clear percept of directional motion, yet Fourier analysis reveals a lack of coherent motion energy, or energy present in a direction opposing that of the displacement (hence the term 'drift-balanced'). Examples of DBS include image sequences in which the contrast polarity of edges present reverses between frames.

A subset of DBS, which are also processed by the network, are known as micro-balanced stimuli (MBS). MBS contain no correlatable features and are drift-balanced at all scales. The MBS image sequences used for this work were created from a random-dot image in which an area is successively shifted by a constant displacement between each frame and *simultaneously* re-randomised.

## 3   EXISTING METHODS OF MOTION DETECTION

### 3.1   CORRELATION METHODS

Correlation methods perform a local cross-correlation in image space: the matching of features in local neighbourhoods (depending upon displacement/speed) between image frames underlies the motion detection. Examples of this method include [Van Santen 1985]. Most correlation models suffer from noise degradation in that any noise features extracted by the edge detection are available for spurious correlation.

There has been much recent debate questioning the validity of correlation methods for modelling human motion detection abilities. In addition to DBS, there is also increasing psychophysical evidence ([Landy 1991, Mather 1991]) which correlation methods cannot account for.

These factors suggest that correlation techniques are not suitable for low-level motion processing where no information is available concerning *what* is moving (as with MBS). However, correlation is a more plausible method when working with higher level constructs such as tracking in model-based vision (e.g. [Bray 1990]).

### 3.2   GRADIENT-FILTER (GF) METHODS

GF methods use a combination of spatial filtering to determine edge positions and temporal filtering to determine whether such edges are moving. A common assumption used by GF methods is that *spatial gradients are constant*. A recent method by Verri [Verri 1990], for example, argues that flow detection is based upon the notion

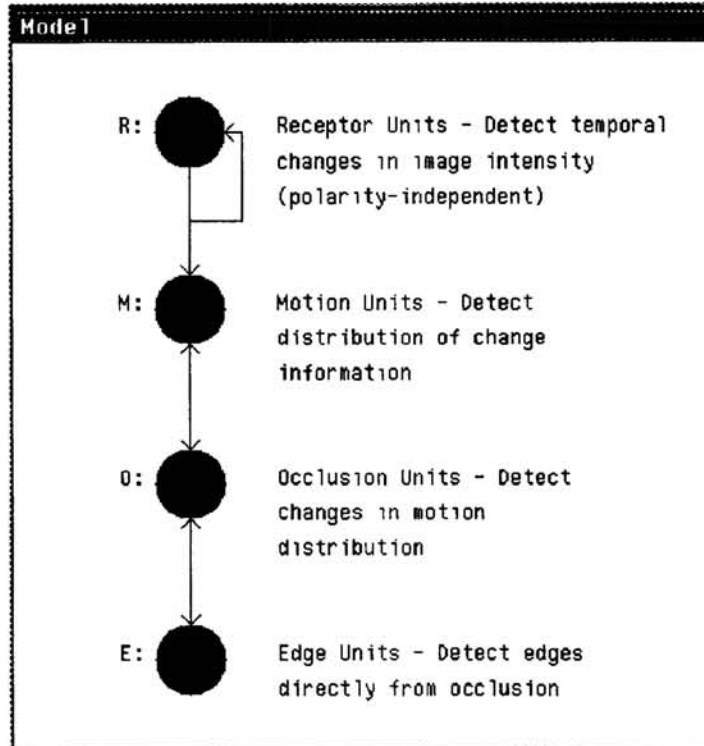

Figure 1: The Network (Schematic)

of tracking spatial gradient magnitude and/or direction, and that any variation in the spatial gradient is due to some form of motion deformation -- i.e. rotation, expansion or shear. Whilst for scenes containing smooth surfaces this is a valid approximation, it is *not* the case for second-order stimuli such as DBS.

## 4   THE NETWORK

A simplified diagram illustrating the basic structure of the network (based upon earlier work ([Tunley 1990, Tunley 1991a, Tunley 1991b]) is shown in Figure 1 (the edge detection stage is discussed elsewhere ([Tunley 1990, Tunley 1991b, Tunley 1992]).

### 4.1   INPUT RECEPTOR UNITS

The units in the input layer respond to rectified local changes in image intensity over time. Each unit has a variable adaption rate, resulting in temporal sensitivity – a fast adaption rate gives a high temporal filtering rate. The main advantages for this temporal averaging processing are:

- Averaging removes the D.C. component of image intensity.   This elimi-nates problematic gain for motion in high brightness areas of the image. [Heeger 1988].

- The random nature of DBS/MBS generation cannot guarantee that each pixel change is due to local image motion.  Local temporal averaging smooths the

moving regions, thus creating a more coherently structured input for the motion units.

The input units have a pointwise rectifying response governed by an autoregressive filter of the following form:

$$R_n = (1 - \alpha).R_{n-1} + \alpha.|I_n - I_{n-1}| \tag{1}$$

where $\alpha \in [0, 1]$ is a variable which controls the degree of temporal filtering of the change in input intensity, $n$ and $n - 1$ are successive image frames, and $R_n$ and $I_n$ are the filter output and input, respectively.

The receptor unit responses for two different $\alpha$ values are shown in Figure 2. $\alpha$ can thus be used to alter the amount of motion blur produced for a particular frame rate, effectively producing a unit with differing velocity sensitivity.

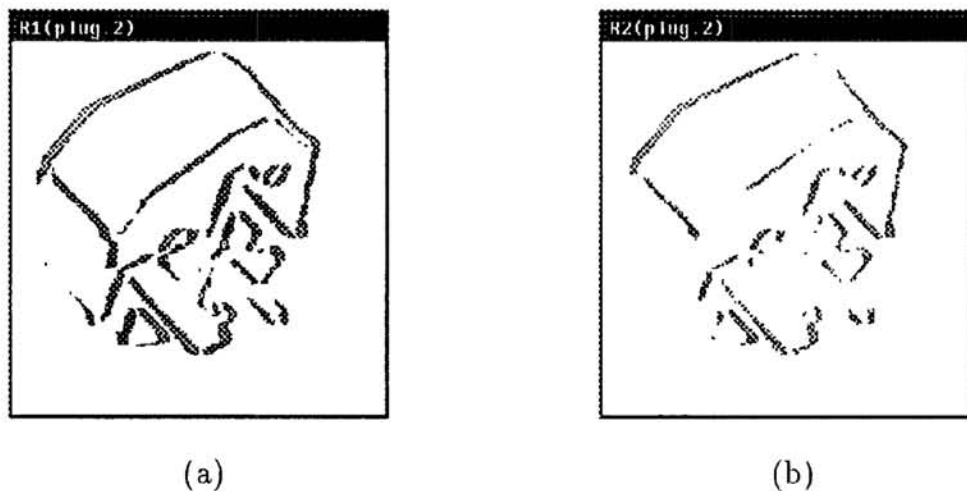

(a)                                    (b)

Figure 2: Receptor Unit Response: (a) $\alpha = 0.3$; (b) $\alpha = 0.7$.

## 4.2 MOTION UNITS

These units determine the *coherence* of image changes indicated by corresponding receptor units. First-order motion produces highly-tuned motion activity – i.e. a strong response in a particular direction – whilst second-order motion results in less coherent output.

The operation of a basic motion detector can be described by:

$$M_{ijkdn} = |R_{ijn} - R_{i'j'n-1}| \tag{2}$$

where $M$ is the detector, $(i', j')$ is a point in frame $n$ at a distance $d$ from $(i, j)$, a point in frame $n - 1$, in the direction $k$. Therefore, for coherent motion (i.e. first-order), in direction $k$ at a speed of $d$ units/frame, as $n \to \infty$:

$$M_{ijkdn} \to 0 \tag{3}$$

The convergence of motion activity can be seen using an example. The stimulus sequence used consists of a bar of re-randomising texture moving to the right in front of a leftward moving background with the same texture (i.e. random dots). The bar motion is second-order as it contains no correlatable features, whilst the background consists of a simple first-order shifting of dots between frames. Figures 3, 4 and 5 show two-dimensional images of the leftward motion activity for the stimulus after 3, 4 and 6 frames respectively. The background, which has coherent leftward movement (at speed $d$ units/frame) is gradually reducing to zero whilst the microbalanced rightwards-moving bar, remains active. The fact that a non-zero response is obtained for second-order motion suggests, according to the definition of Chubb and Sperling [Chubb 1989], that first-order detectors produce no response to MBS, that this detector is second-order with regard to motion detection.

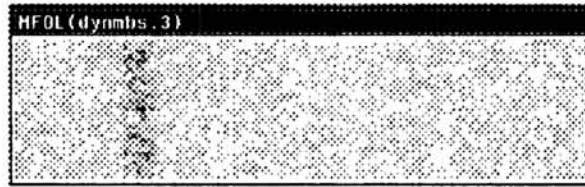

Figure 3: Leftward Motion Response to Third Frame in Sequence.

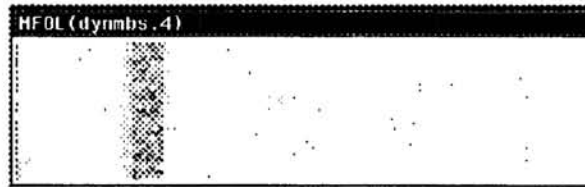

Figure 4: Leftward Motion Response to Fourth Frame.

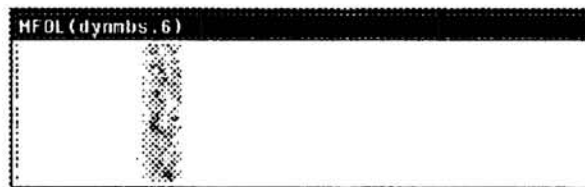

Figure 5: Leftward Motion Response to Sixth Frame.

The motion units in this model are arranged on a hexagonal grid. This grid is known as a flow web as it allows information to flow, both laterally between units of the same type, and between the different units in the model (motion, occlusion or edge). Each flow web unit is represented by three variables – a position $(a, b)$ and a direction $k$, which is evenly spaced between 0 and 360 degrees. In this model each $k$ is an integer between 1 and $k_{\mathrm{max}}$ – the value of $k_{\mathrm{max}}$ can be varied to vary the sensitivity of the units.

A way of using first-order techniques to discriminate between first and second-order motions is through the concept of coherence. At any point in the motion-processed images in Figures 3-5, a measure of the overall variation in motion activity can be used to distinguish between the motion of the micro-balanced bar and its background. The motion energy for a detector with displacement $d$, and orientation

$k$, at position $(a, b)$, can be represented by $E_{abkd}$. For each motion unit, responding over distance $d$, in each cluster the energy present can be defined as:

$$E_{abkdn} = \frac{\min_k (M_{abkd})}{M_{abkd}} \tag{4}$$

where $\min_k(x_k)$ is the minimum value of $x$ found searching over $k$ values. If motion is coherent, and of approximately the correct speed for the detector $M$, then as $n \to \infty$:

$$E_{abk_m dn} \to 1 \tag{5}$$

where $k_m$ is in the actual direction of the motion. In reality $n$ need only approach around 5 for convergence to occur. Also, more importantly, under the same convergence conditions:

$$E_{abkdn} \to 0 \ \forall k \neq k_m \tag{6}$$

This is due to the fact that the minimum activation value in a group of first-order detectors at point $(a, b)$ will be the same as the actual value in the direction, $k_m$. By similar reasoning, for non-coherent motion as $n \to \infty$:

$$E_{abkdn} \to 1 \ \forall k \tag{7}$$

in other words there is no peak of activity in a given direction. The motion energy is ambiguous at a large number of points in most images, except at discontinuities and on well-textured surfaces.

A measure of motion coherence used for the motion units can now be defined as:

$$M_c(abkd) = \frac{E_{abkd}}{\sum_{k=1}^{k_{max}} E_{abkd}} \tag{8}$$

For coherent motion in direction $k_m$ as $n \to \infty$:

$$M_c(abk_m d) \to 1 \tag{9}$$

Whilst for second-order motion, also as $n \to \infty$:

$$M_c(abkd) \to 1/k_{max} \ \forall k \tag{10}$$

Using this approach the total $M_c$ activity at each position – regardless of coherence, or lack of it – is unity. Motion energy is the same in all moving regions, the difference is in the distribution, or *tuning* of that energy.

Figures 6, 7 and 8 show how motion coherence allows the flow web structure to reveal the presence of motion in microbalanced areas whilst not affecting the easily detected background motion for the stimulus.

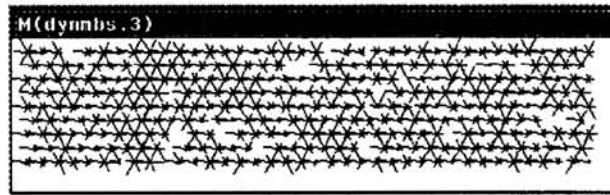

Figure 6: Motion Coherence Response to Third Frame

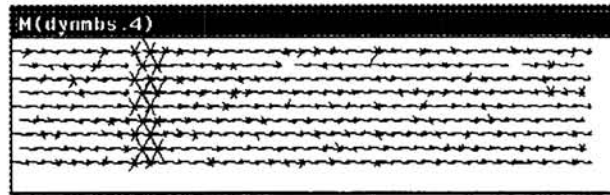

Figure 7: Motion Coherence Response to Fourth Frame

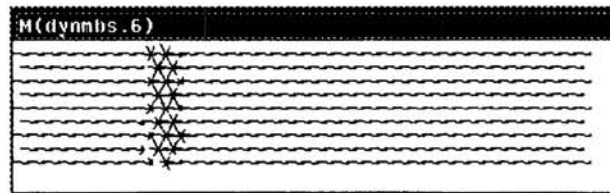

Figure 8: Motion Coherence Response to Sixth Frame

## 4.3   OCCLUSION UNITS

These units identify discontinuities in second-order motion which are vitally important when computing the direction of that motion. They determine spatial and temporal changes in motion coherence and can process single or multiple motions at each image point. Established and newly-activated occlusion units work, through a gating process, to enhance continuously-displacing surfaces, utilising the concept of visual inertia.

The implementation details of the occlusion stage of this model are discussed elsewhere [Tunley 1992], but some output from the occlusion units to the above second-order stimulus are shown in Figures 9 and 10. The figures show how the edges of the bar can be determined.

## Footnotes

\*Current address: Experimental Psychology, School of Biological Sciences. Sussex University.

# References

[Adelson 1985]    E.H. Adelson and J.R. Bergen. Spatiotemporal energy models for the perception of motion. *J. Opt. Soc. Am.* 2, 1985.

[Bray 1990]    A.J. Bray. Tracking objects using image disparities. *Image and Vision Computing*, 8, 1990.

[Chubb 1989]    C. Chubb and G. Sperling. Second-order motion perception: Space/time separable mechanisms. In *Proc. Workshop on Visual Motion, Irvine, CA, USA*, 1989.

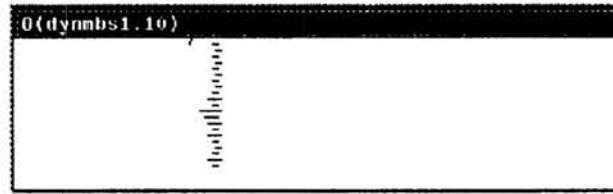

Figure 9: Occluding Motion Information: Occlusion activity produced by an increase in motion coherence activity.

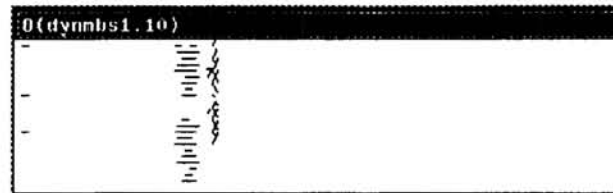

Figure 10: Occluding Motion Information: Occlusion activity produced by a decrease in motion activity at a point. Some spurious activity is produced due to the random nature of the second-order motion information.

[Heeger 1988]        D.J. Heeger. Optical Flow using spatiotemporal filters. *Int. J. Comp. Vision*, 1, 1988.

[Koffka 1935]        K. Koffka. *Principles of Gestalt Psychology*. Harcourt Brace, 1935.

[Landy 1991]         M.S. Landy, B.A. Dosher, G. Sperling and M.E. Perkins. The kinetic depth effect and optic flow II: First- and second-order motion. *Vis. Res.* 31, 1991.

[Mather 1991]        G. Mather. Personal Communication.

[Reichardt 1961]     W. Reichardt. Autocorrelation, a principle for the evaluation of sensory information by the central nervous system. In W. Rosenblith, editor, *Sensory Communications*. Wiley NY, 1961.

[Van Santen 1985]    J.P.H. Van Santen and G. Sperling. Elaborated Reichardt detectors. *J. Opt. Soc. Am.* 2, 1985.

[Tunley 1990]        H. Tunley. Segmenting Moving Images. In *Proc. Int. Neural Network Conf. (INNC90), Paris, France*, 1990.

[Tunley 1991a]       H. Tunley. Distributed dynamic processing for edge detection. In *Proc. British Machine Vision Conf. (BMVC91), Glasgow, Scotland*, 1991.

[Tunley 1991b]       H. Tunley. Dynamic segmentation and optic flow extraction. In *Proc. Int. Joint. Conf. Neural Networks (IJCNN91), Seattle, USA*, 1991.

[Tunley 1992]        H. Tunley. Sceond-order motion processing: A distributed approach. CSRP 211, School of Cognitive and Computing Sciences, University of Sussex (forthcoming).

[Verri 1990]         A. Verri, F. Girosi and V. Torre. Differential techniques for optic flow. *J. Opt. Soc. Am.* 7, 1990.
